# The Tradeoffs of Large Scale Learning

**Léon Bottou**
NEC laboratories of America
Princeton, NJ 08540, USA
`leon@bottou.org`

**Olivier Bousquet**
Google Zürich
8002 Zurich, Switzerland
`olivier.bousquet@m4x.org`

## Abstract

This contribution develops a theoretical framework that takes into account the effect of approximate optimization on learning algorithms. The analysis shows distinct tradeoffs for the case of small-scale and large-scale learning problems. Small-scale learning problems are subject to the usual approximation–estimation tradeoff. Large-scale learning problems are subject to a qualitatively different tradeoff involving the computational complexity of the underlying optimization algorithms in non-trivial ways.

## 1 Motivation

The computational complexity of learning algorithms has seldom been taken into account by the learning theory. Valiant [1] states that a problem is "learnable" when there exists a probably approximatively correct learning algorithm *with polynomial complexity*. Whereas much progress has been made on the statistical aspect (e.g., [2, 3, 4]), very little has been told about the complexity side of this proposal (e.g., [5].)

Computational complexity becomes the limiting factor when one envisions large amounts of training data. Two important examples come to mind:

- Data mining exists because competitive advantages can be achieved by analyzing the masses of data that describe the life of our computerized society. Since virtually every computer generates data, the data volume is proportional to the available computing power. Therefore one needs learning algorithms that scale roughly linearly with the total volume of data.

- Artificial intelligence attempts to emulate the cognitive capabilities of human beings. Our biological brains can learn quite efficiently from the continuous streams of perceptual data generated by our six senses, using limited amounts of sugar as a source of power. This observation suggests that there are learning algorithms whose computing time requirements scale roughly linearly with the total volume of data.

This contribution finds its source in the idea that approximate optimization algorithms might be sufficient for learning purposes. The first part proposes new decomposition of the test error where an additional term represents the impact of approximate optimization. In the case of small-scale learning problems, this decomposition reduces to the well known tradeoff between approximation error and estimation error. In the case of large-scale learning problems, the tradeoff is more complex because it involves the computational complexity of the learning algorithm. The second part explores the asymptotic properties of the large-scale learning tradeoff for various prototypical learning algorithms under various assumptions regarding the statistical estimation rates associated with the chosen objective functions. This part clearly shows that the best optimization algorithms are not necessarily the best learning algorithms. Maybe more surprisingly, certain algorithms perform well regardless of the assumed rate for the statistical estimation error.

## 2 Approximate Optimization

### 2.1 Setup

Following [6, 2], we consider a space of input-output pairs $(x, y) \in \mathcal{X} \times \mathcal{Y}$ endowed with a probability distribution $P(x, y)$. The conditional distribution $P(y|x)$ represents the unknown relationship between inputs and outputs. The discrepancy between the predicted output $\hat{y}$ and the real output $y$ is measured with a loss function $\ell(\hat{y}, y)$. Our benchmark is the function $f^*$ that minimizes the expected risk

$$E(f) = \int \ell(f(x), y) \, dP(x, y) = \mathbb{E}\left[\ell(f(x), y)\right],$$

that is,

$$f^*(x) = \arg\min_{\hat{y}} \mathbb{E}\left[\ell(\hat{y}, y)| \, x\right].$$

Although the distribution $P(x, y)$ is unknown, we are given a sample $\mathcal{S}$ of $n$ independently drawn training examples $(x_i, y_i)$, $i = 1 \dots n$. We define the empirical risk

$$E_n(f) = \frac{1}{n}\sum_{i=1}^{n} \ell(f(x_i), y_i) = \mathbb{E}_n[\ell(f(x), y)].$$

Our first learning principle consists in choosing a family $\mathcal{F}$ of candidate prediction functions and finding the function $f_n = \arg\min_{f \in \mathcal{F}} E_n(f)$ that minimizes the empirical risk. Well known combinatorial results (e.g., [2]) support this approach provided that the chosen family $\mathcal{F}$ is sufficiently restrictive. Since the optimal function $f^*$ is unlikely to belong to the family $\mathcal{F}$, we also define $f_{\mathcal{F}}^* = \arg\min_{f \in \mathcal{F}} E(f)$. For simplicity, we assume that $f^*$, $f_{\mathcal{F}}^*$ and $f_n$ are well defined and unique.

We can then decompose the excess error as

$$\mathbb{E}\left[E(f_n) - E(f^*)\right] = \mathbb{E}\left[E(f_{\mathcal{F}}^*) - E(f^*)\right] + \mathbb{E}\left[E(f_n) - E(f_{\mathcal{F}}^*)\right] = \mathcal{E}_{\text{app}} + \mathcal{E}_{\text{est}}, \quad (1)$$

where the expectation is taken with respect to the random choice of training set. The *approximation error* $\mathcal{E}_{\text{app}}$ measures how closely functions in $\mathcal{F}$ can approximate the optimal solution $f^*$. The *estimation error* $\mathcal{E}_{\text{est}}$ measures the effect of minimizing the empirical risk $E_n(f)$ instead of the expected risk $E(f)$. The estimation error is determined by the number of training examples and by the capacity of the family of functions [2]. Large families[1] of functions have *smaller approximation errors* but lead to *higher estimation errors*. This tradeoff has been extensively discussed in the literature [2, 3] and lead to excess error that scale between the inverse and the inverse square root of the number of examples [7, 8].

### 2.2 Optimization Error

Finding $f_n$ by minimizing the empirical risk $E_n(f)$ is often a computationally expensive operation. Since the empirical risk $E_n(f)$ is already an approximation of the expected risk $E(f)$, it should not be necessary to carry out this minimization with great accuracy. For instance, we could stop an iterative optimization algorithm long before its convergence.

Let us assume that our minimization algorithm returns an approximate solution $\tilde{f}_n$ such that

$$E_n(\tilde{f}_n) < E_n(f_n) + \rho$$

where $\rho \geq 0$ is a predefined tolerance. An additional term $\mathcal{E}_{\text{opt}} = \mathbb{E}\left[E(\tilde{f}_n) - E(f_n)\right]$ then appears in the decomposition of the excess error $\mathcal{E} = \mathbb{E}\left[E(\tilde{f}_n) - E(f^*)\right]$:

$$\begin{aligned}
\mathcal{E} &= \mathbb{E}\left[E(f_{\mathcal{F}}^*) - E(f^*)\right] + \mathbb{E}\left[E(f_n) - E(f_{\mathcal{F}}^*)\right] + \mathbb{E}\left[E(\tilde{f}_n) - E(f_n)\right] \\
&= \mathcal{E}_{\text{app}} + \mathcal{E}_{\text{est}} + \mathcal{E}_{\text{opt}}.
\end{aligned} \quad (2)$$

We call this additional term *optimization error*. It reflects the impact of the approximate optimization on the generalization performance. Its magnitude is comparable to $\rho$ (see section 3.1.)

## 2.3 The Approximation–Estimation–Optimization Tradeoff

This decomposition leads to a more complicated compromise. It involves three variables and two constraints. The constraints are the maximal number of available training example and the maximal computation time. The variables are the size of the family of functions $\mathcal{F}$, the optimization accuracy $\rho$, and the number of examples $n$. This is formalized by the following optimization problem.

$$\min_{\mathcal{F},\rho,n} \quad \mathcal{E} = \mathcal{E}_{\text{app}} + \mathcal{E}_{\text{est}} + \mathcal{E}_{\text{opt}} \quad \text{subject to} \left\{ \begin{array}{ccc} n & \leq & n_{\max} \\ T(\mathcal{F},\rho,n) & \leq & T_{\max} \end{array} \right. \tag{3}$$

The number $n$ of training examples is a variable because we could choose to use only a subset of the available training examples in order to complete the optimization within the alloted time. This happens often in practice. Table 1 summarizes the typical evolution of the quantities of interest with the three variables $\mathcal{F}$, $n$, and $\rho$ increase.

Table 1: Typical variations when $\mathcal{F}$, $n$, and $\rho$ increase.

|  |  | $\mathcal{F}$ | $n$ | $\rho$ |
|---|---|---|---|---|
| $\mathcal{E}_{\text{app}}$ | (approximation error) | ↘ | | |
| $\mathcal{E}_{\text{est}}$ | (estimation error) | ↗ | ↘ | |
| $\mathcal{E}_{\text{opt}}$ | (optimization error) | $\cdots$ | $\cdots$ | ↗ |
| $T$ | (computation time) | ↗ | ↗ | ↘ |

The solution of the optimization program (3) depends critically of which budget constraint is active: constraint $n < n_{\max}$ on the number of examples, or constraint $T < T_{\max}$ on the training time.

- We speak of *small-scale learning problem* when (3) is constrained by the maximal number of examples $n_{\max}$. Since the computing time is not limited, we can reduce the optimization error $\mathcal{E}_{\text{opt}}$ to insignificant levels by choosing $\rho$ arbitrarily small. The excess error is then dominated by the approximation and estimation errors, $\mathcal{E}_{\text{app}}$ and $\mathcal{E}_{\text{est}}$. Taking $n = n_{\max}$, we recover the approximation-estimation tradeoff that is the object of abundant literature.

- We speak of *large-scale learning problem* when (3) is constrained by the maximal computing time $T_{\max}$. Approximate optimization, that is choosing $\rho > 0$, possibly can achieve better generalization because more training examples can be processed during the allowed time. The specifics depend on the computational properties of the chosen optimization algorithm through the expression of the computing time $T(\mathcal{F},\rho,n)$.

## 3 The Asymptotics of Large-scale Learning

In the previous section, we have extended the classical approximation-estimation tradeoff by taking into account the optimization error. We have given an objective criterion to distiguish small-scale and large-scale learning problems. In the small-scale case, we recover the classical tradeoff between approximation and estimation. The large-scale case is substantially different because it involves the computational complexity of the learning algorithm. In order to clarify the large-scale learning tradeoff with sufficient generality, this section makes several simplifications:

- We are studying upper bounds of the approximation, estimation, and optimization errors (2). It is often accepted that these upper bounds give a realistic idea of the actual convergence rates [9, 10, 11, 12]. Another way to find comfort in this approach is to say that we study guaranteed convergence rates instead of the possibly pathological special cases.

- We are studying the asymptotic properties of the tradeoff when the problem size increases. Instead of carefully balancing the three terms, we write $\mathcal{E} = \mathcal{O}(\mathcal{E}_{\text{app}}) + \mathcal{O}(\mathcal{E}_{\text{est}}) + \mathcal{O}(\mathcal{E}_{\text{opt}})$ and only need to ensure that the three terms decrease with the same asymptotic rate.

- We are considering a fixed family of functions $\mathcal{F}$ and therefore avoid taking into account the approximation error $\mathcal{E}_{\text{app}}$. This part of the tradeoff covers a wide spectrum of practical realities such as choosing models and choosing features. In the context of this work, we do

not believe we can meaningfully address this without discussing, for instance, the thorny issue of feature selection. Instead we focus on the choice of optimization algorithm.

- Finally, in order to keep this paper short, we consider that the family of functions $\mathcal{F}$ is linearly parametrized by a vector $w \in \mathbb{R}^d$. We also assume that $x$, $y$ and $w$ are bounded, ensuring that there is a constant B such that $0 \leq \ell(f_w(x), y) \leq B$ and $\ell(\cdot, y)$ is Lipschitz.

We first explain how the uniform convergence bounds provide convergence rates that take the optimization error into account. Then we discuss and compare the asymptotic learning properties of several optimization algorithms.

## 3.1 Convergence of the Estimation and Optimization Errors

The optimization error $\mathcal{E}_{\mathrm{opt}}$ depends directly on the optimization accuracy $\rho$. However, the accuracy $\rho$ involves the empirical quantity $E_n(\tilde{f}_n) - E_n(f_n)$, whereas the optimization error $\mathcal{E}_{\mathrm{opt}}$ involves its expected counterpart $E(\tilde{f}_n) - E(f_n)$. This section discusses the impact on the optimization error $\mathcal{E}_{\mathrm{opt}}$ and of the optimization accuracy $\rho$ on generalization bounds that leverage the uniform convergence concepts pioneered by Vapnik and Chervonenkis (e.g., [2].)

In this discussion, we use the letter $c$ to refer to any positive constant. Multiple occurences of the letter $c$ do not necessarily imply that the constants have identical values.

### 3.1.1 Simple Uniform Convergence Bounds

Recall that we assume that $\mathcal{F}$ is linearly parametrized by $w \in \mathbb{R}^d$. Elementary uniform convergence results then state that

$$\mathbb{E}\left[\sup_{f \in \mathcal{F}} |E(f) - E_n(f)|\right] \leq c\sqrt{\frac{d}{n}},$$

where the expectation is taken with respect to the random choice of the training set.[2] This result immediately provides a bound on the estimation error:

$$\begin{aligned}
\mathcal{E}_{\mathrm{est}} &= \mathbb{E}\left[\left(E(f_n) - E_n(f_n)\right) + \left(E_n(f_n) - E_n(f_{\mathcal{F}}^*)\right) + \left(E_n(f_{\mathcal{F}}^*) - E(f_{\mathcal{F}}^*)\right)\right] \\
&\leq 2\,\mathbb{E}\left[\sup_{f \in \mathcal{F}} |E(f) - E_n(f)|\right] \leq c\sqrt{\frac{d}{n}}.
\end{aligned}$$

This same result also provides a combined bound for the estimation and optimization errors:

$$\begin{aligned}
\mathcal{E}_{\mathrm{est}} + \mathcal{E}_{\mathrm{opt}} &= \mathbb{E}\left[E(\tilde{f}_n) - E_n(\tilde{f}_n)\right] + \mathbb{E}\left[E_n(\tilde{f}_n) - E_n(f_n)\right] \\
&+ \mathbb{E}\left[E_n(f_n) - E_n(f_{\mathcal{F}}^*)\right] + \mathbb{E}\left[E_n(f_{\mathcal{F}}^*) - E(f_{\mathcal{F}}^*)\right] \\
&\leq c\sqrt{\frac{d}{n}} + \rho + 0 + c\sqrt{\frac{d}{n}} = c\left(\rho + \sqrt{\frac{d}{n}}\right).
\end{aligned}$$

Unfortunately, this convergence rate is known to be pessimistic in many important cases. More sophisticated bounds are required.

### 3.1.2 Faster Rates in the Realizable Case

When the loss functions $\ell(\hat{y}, y)$ is positive, with probability $1 - e^{-\tau}$ for any $\tau > 0$, relative uniform convergence bounds state that

$$\sup_{f \in \mathcal{F}} \frac{E(f) - E_n(f)}{\sqrt{E(f)}} \leq c\sqrt{\frac{d}{n}\log\frac{n}{d} + \frac{\tau}{n}}.$$

This result is very useful because it provides faster convergence rates $\mathcal{O}(\log n/n)$ in the *realizable case*, that is when $\ell(f_n(x_i), y_i) = 0$ for all training examples $(x_i, y_i)$. We have then $E_n(f_n) = 0$, $E_n(\tilde{f}_n) \leq \rho$, and we can write

$$E(\tilde{f}_n) - \rho \leq c\sqrt{E(\tilde{f}_n)}\sqrt{\frac{d}{n}\log\frac{n}{d} + \frac{\tau}{n}}.$$

Viewing this as a second degree polynomial inequality in variable $\sqrt{E(\tilde{f}_n)}$, we obtain

$$E(\tilde{f}_n) \leq c \left( \rho + \frac{d}{n} \log \frac{n}{d} + \frac{\tau}{n} \right) .$$

Integrating this inequality using a standard technique (see, e.g., [13]), we obtain a better convergence rate of the combined estimation and optimization error:

$$\mathcal{E}_{\text{est}} + \mathcal{E}_{\text{opt}} = \mathbb{E}\left[ E(\tilde{f}_n) - E(f_{\mathcal{F}}^*) \right] \leq \mathbb{E}\left[ E(\tilde{f}_n) \right] = c \left( \rho + \frac{d}{n} \log \frac{n}{d} \right) .$$

### 3.1.3   Fast Rate Bounds

Many authors (e.g., [10, 4, 12]) obtain fast statistical estimation rates in more general conditions. These bounds have the general form

$$\mathcal{E}_{\text{app}} + \mathcal{E}_{\text{est}} \leq c \left( \mathcal{E}_{\text{app}} + \left( \frac{d}{n} \log \frac{n}{d} \right)^{\alpha} \right) \quad \text{for } \frac{1}{2} \leq \alpha \leq 1 . \tag{4}$$

This result holds when one can establish the following variance condition:

$$\forall f \in \mathcal{F} \quad \mathbb{E}\left[ \left( \ell(f(X), Y) - \ell(f_{\mathcal{F}}^*(X), Y) \right)^2 \right] \leq c \left( E(f) - E(f_{\mathcal{F}}^*) \right)^{2 - \frac{1}{\alpha}} . \tag{5}$$

The convergence rate of (4) is described by the exponent $\alpha$ which is determined by the quality of the variance bound (5). Works on fast statistical estimation identify two main ways to establish such a variance condition.

- Exploiting the strict convexity of certain loss functions [12, theorem 12]. For instance, Lee et al. [14] establish a $\mathcal{O}(\log n/n)$ rate using the squared loss $\ell(\hat{y}, y) = (\hat{y} - y)^2$.
- Making assumptions on the data distribution. In the case of pattern recognition problems, for instance, the "Tsybakov condition" indicates how cleanly the posterior distributions $P(y|x)$ cross near the optimal decision boundary [11, 12]. The realizable case discussed in section 3.1.2 can be viewed as an extreme case of this.

Despite their much greater complexity, fast rate estimation results can accomodate the optimization accuracy $\rho$ using essentially the methods illustrated in sections 3.1.1 and 3.1.2. We then obtain a bound of the form

$$\mathcal{E} = \mathcal{E}_{\text{app}} + \mathcal{E}_{\text{est}} + \mathcal{E}_{\text{opt}} = \mathbb{E}\left[ E(\tilde{f}_n) - E(f^*) \right] \leq c \left( \mathcal{E}_{\text{app}} + \left( \frac{d}{n} \log \frac{n}{d} \right)^{\alpha} + \rho \right) . \tag{6}$$

For instance, a general result with $\alpha = 1$ is provided by Massart [13, theorem 4.2]. Combining this result with standard bounds on the complexity of classes of linear functions (e.g., [10]) yields the following result:

$$\mathcal{E} = \mathcal{E}_{\text{app}} + \mathcal{E}_{\text{est}} + \mathcal{E}_{\text{opt}} = \mathbb{E}\left[ E(\tilde{f}_n) - E(f^*) \right] \leq c \left( \mathcal{E}_{\text{app}} + \frac{d}{n} \log \frac{n}{d} + \rho \right) . \tag{7}$$

See also [15, 4] for more bounds taking into account the optimization accuracy.

### 3.2   Gradient Optimization Algorithms

We now discuss and compare the asymptotic learning properties of four gradient optimization algorithms. Recall that the family of function $\mathcal{F}$ is linearly parametrized by $w \in \mathbb{R}^d$. Let $w_{\mathcal{F}}^*$ and $w_n$ correspond to the functions $f_{\mathcal{F}}^*$ and $f_n$ defined in section 2.1. In this section, we assume that the functions $w \mapsto \ell(f_w(x), y)$ are convex and twice differentiable with continuous second derivatives. Convexity ensures that the empirical const function $C(w) = E_n(f_w)$ has a single minimum.

Two matrices play an important role in the analysis: the Hessian matrix $H$ and the gradient covariance matrix $G$, both measured at the empirical optimum $w_n$.

$$H = \frac{\partial^2 C}{\partial w^2}(w_n) = \mathbb{E}_n\left[ \frac{\partial^2 \ell(f_{w_n}(x), y)}{\partial w^2} \right] , \tag{8}$$

$$G = \mathbb{E}_n\left[ \left( \frac{\partial \ell(f_{w_n}(x), y)}{\partial w} \right) \left( \frac{\partial \ell(f_{w_n}(x), y)}{\partial w} \right)' \right] . \tag{9}$$

The relation between these two matrices depends on the chosen loss function. In order to summarize them, we assume that there are constants $\lambda_{\max} \geq \lambda_{\min} > 0$ and $\nu > 0$ such that, for any $\eta > 0$, we can choose the number of examples $n$ large enough to ensure that the following assertion is true with probability greater than $1 - \eta$ :

$$\text{tr}(G\, H^{-1}) \leq \nu \qquad \text{and} \qquad \text{EigenSpectrum}(H) \subset [\,\lambda_{\min}\,,\,\lambda_{\max}\,] \qquad (10)$$

The condition number $\kappa = \lambda_{\max}/\lambda_{\min}$ is a good indicator of the difficulty of the optimization [16].

The condition $\lambda_{\min} > 0$ avoids complications with stochastic gradient algorithms. Note that this condition only implies strict convexity around the optimum. For instance, consider the loss function $\ell$ is obtained by smoothing the well known hinge loss $\ell(z, y) = \max\{0, 1 - yz\}$ in a small neighborhood of its non-differentiable points. Function $C(w)$ is then piecewise linear with smoothed edges and vertices. It is not strictly convex. However its minimum is likely to be on a smoothed vertex with a non singular Hessian. When we have strict convexity, the argument of [12, theorem 12] yields fast estimation rates $\alpha \approx 1$ in (4) and (6). This is not necessarily the case here.

The four algorithm considered in this paper use information about the gradient of the cost function to iteratively update their current estimate $w(t)$ of the parameter vector.

- **Gradient Descent (GD)** iterates

$$w(t+1) \;=\; w(t) - \eta \frac{\partial C}{\partial w}(w(t)) \;=\; w(t) - \eta \frac{1}{n} \sum_{i=1}^{n} \frac{\partial}{\partial w} \ell\big(f_{w(t)}(x_i), y_i\big)$$

  where $\eta > 0$ is a small enough gain. GD is an algorithm with linear convergence [16]. When $\eta = 1/\lambda_{\max}$, this algorithm requires $\mathcal{O}(\kappa \log(1/\rho))$ iterations to reach accuracy $\rho$. The exact number of iterations depends on the choice of the initial parameter vector.

- **Second Order Gradient Descent (2GD)** iterates

$$w(t+1) \;=\; w(t) - H^{-1} \frac{\partial C}{\partial w}(w(t)) \;=\; w(t) - \frac{1}{n} H^{-1} \sum_{i=1}^{n} \frac{\partial}{\partial w} \ell\big(f_{w(t)}(x_i), y_i\big)$$

  where matrix $H^{-1}$ is the inverse of the Hessian matrix (8). This is more favorable than Newton's algorithm because we do not evaluate the local Hessian at each iteration but simply assume that we know in advance the Hessian at the optimum. 2GD is a superlinear optimization algorithm with quadratic convergence [16]. When the cost is quadratic, a single iteration is sufficient. In the general case, $\mathcal{O}(\log\log(1/\rho))$ iterations are required to reach accuracy $\rho$.

- **Stochastic Gradient Descent (SGD)** picks a random training example $(x_t, y_t)$ at each iteration and updates the parameter $w$ on the basis of this example only,

$$w(t+1) \;=\; w(t) - \frac{\eta}{t} \frac{\partial}{\partial w} \ell\big(f_{w(t)}(x_t), y_t\big).$$

  Murata [17, section 2.2], characterizes the mean $\mathbb{E}_s[w(t)]$ and variance $\mathbb{V}\text{ar}_s[w(t)]$ with respect to the distribution implied by the random examples drawn from the training set $\mathcal{S}$ at each iteration. Applying this result to the discrete training set distribution for $\eta = 1/\lambda_{\min}$, we have $\delta w(t)^2 = \mathcal{O}(1/t)$ where $\delta w(t)$ is a shorthand notation for $w(t) - w_n$. We can then write

$$\begin{aligned} \mathbb{E}_{\mathcal{S}}[\,C(w(t)) - \inf C\,] \;&=\; \mathbb{E}_{\mathcal{S}}\big[\text{tr}\big(H\,\delta w(t)\,\delta w(t)'\big)\big] + \text{o}\big(\tfrac{1}{t}\big) \\ &=\; \text{tr}\big(H\,\mathbb{E}_{\mathcal{S}}[\delta w(t)]\,\mathbb{E}_{\mathcal{S}}[\delta w(t)]' + H\,\mathbb{V}\text{ar}_{\mathcal{S}}[w(t)]\big) + \text{o}\big(\tfrac{1}{t}\big) \qquad (11) \\ &\leq\; \tfrac{\text{tr}(GH)}{t} + \text{o}\big(\tfrac{1}{t}\big) \;\leq\; \tfrac{\nu \kappa^2}{t} + \text{o}\big(\tfrac{1}{t}\big). \end{aligned}$$

  Therefore the SGD algorithm reaches accuracy $\rho$ after less than $\nu\kappa^2/\rho + \text{o}(1/\rho)$ iterations on average. The SGD convergence is essentially limited by the stochastic noise induced by the random choice of one example at each iteration. Neither the initial value of the parameter vector $w$ nor the total number of examples $n$ appear in the dominant term of this bound! When the training set is large, one could reach the desired accuracy $\rho$ measured on the whole training set without even visiting all the training examples. This is in fact a kind of generalization bound.

Table 2: Asymptotic results for gradient algorithms (with probability 1). Compare the second last column (time to optimize) with the last column (time to reach the excess test error $\epsilon$). *Legend*: $n$ number of examples; $d$ parameter dimension; $\kappa$, $\nu$ see equation (10).

| Algorithm | Cost of one iteration | Iterations to reach $\rho$ | Time to reach accuracy $\rho$ | Time to reach $\mathcal{E} \leq c\left(\mathcal{E}_{\mathrm{app}} + \varepsilon\right)$ |
|:---:|:---:|:---:|:---:|:---:|
| GD | $\mathcal{O}(nd)$ | $\mathcal{O}\!\left(\kappa \log \frac{1}{\rho}\right)$ | $\mathcal{O}\!\left(nd\kappa \log \frac{1}{\rho}\right)$ | $\mathcal{O}\!\left(\frac{d^2 \kappa}{\varepsilon^{1/\alpha}} \log^2 \frac{1}{\varepsilon}\right)$ |
| 2GD | $\mathcal{O}(d^2 + nd)$ | $\mathcal{O}\!\left(\log\log \frac{1}{\rho}\right)$ | $\mathcal{O}\!\left((d^2 + nd)\log\log \frac{1}{\rho}\right)$ | $\mathcal{O}\!\left(\frac{d^2}{\varepsilon^{1/\alpha}} \log \frac{1}{\varepsilon} \log\log \frac{1}{\varepsilon}\right)$ |
| SGD | $\mathcal{O}(d)$ | $\frac{\nu\kappa^2}{\rho} + \mathrm{o}\!\left(\frac{1}{\rho}\right)$ | $\mathcal{O}\!\left(\frac{d\nu\kappa^2}{\rho}\right)$ | $\mathcal{O}\!\left(\frac{d\,\nu\,\kappa^2}{\varepsilon}\right)$ |
| 2SGD | $\mathcal{O}(d^2)$ | $\frac{\nu}{\rho} + \mathrm{o}\!\left(\frac{1}{\rho}\right)$ | $\mathcal{O}\!\left(\frac{d^2\nu}{\rho}\right)$ | $\mathcal{O}\!\left(\frac{d^2\,\nu}{\varepsilon}\right)$ |

- **Second Order Stochastic Gradient Descent (2SGD)** replaces the gain $\eta$ by the inverse of the Hessian matrix $H$:

$$w(t+1) \;=\; w(t) - \frac{1}{t}\, H^{-1}\, \frac{\partial}{\partial w} \ell\big(f_{w(t)}(x_t), y_t\big).$$

  Unlike standard gradient algorithms, using the second order information does not change the influence of $\rho$ on the convergence rate but improves the constants. Using again [17, theorem 4], accuracy $\rho$ is reached after $\nu/\rho + \mathrm{o}(1/\rho)$ iterations.

For each of the four gradient algorithms, the first three columns of table 2 report the time for a single iteration, the number of iterations needed to reach a predefined accuracy $\rho$, and their product, the time needed to reach accuracy $\rho$. These asymptotic results are valid with probability 1, since the probability of their complement is smaller than $\eta$ for any $\eta > 0$.

The fourth column bounds the time necessary to reduce the excess error $\mathcal{E}$ below $c\left(\mathcal{E}_{\mathrm{app}} + \varepsilon\right)$ where $c$ is the constant from (6). This is computed by observing that choosing $\rho \sim \left(\frac{d}{n} \log \frac{n}{d}\right)^{\alpha}$ in (6) achieves the fastest rate for $\varepsilon$, with minimal computation time. We can then use the asymptotic equivalences $\rho \sim \varepsilon$ and $n \sim \frac{d}{\varepsilon^{1/\alpha}} \log \frac{1}{\varepsilon}$. Setting the fourth column expressions to $T_{\max}$ and solving for $\epsilon$ yields the *best excess error achieved by each algorithm* within the limited time $T_{\max}$. This provides the asymptotic solution of the Estimation–Optimization tradeoff (3) for large scale problems satisfying our assumptions.

These results clearly show that the generalization performance of *large-scale learning systems* depends on both the statistical properties of the estimation procedure and the computational properties of the chosen optimization algorithm. Their combination leads to surprising consequences:

- *The SGD and 2SGD results do not depend on the estimation rate $\alpha$.* When the estimation rate is poor, there is less need to optimize accurately. That leaves time to process more examples. A potentially more useful interpretation leverages the fact that (11) is already a kind of generalization bound: its fast rate trumps the slower rate assumed for the estimation error.

- *Second order algorithms bring little asymptotical improvements in $\varepsilon$.* Although the super-linear 2GD algorithm improves the logarithmic term, all four algorithms are dominated by the polynomial term in $(1/\varepsilon)$. However, there are important variations in the influence of the constants $d$, $\kappa$ and $\nu$. These constants are very important in practice.

- *Stochastic algorithms (SGD, 2SGD) yield the best generalization performance despite being the worst optimization algorithms.* This had been described before [18] and observed in experiments.

In contrast, since the optimization error $\mathcal{E}_{\mathrm{opt}}$ of *small-scale learning systems* can be reduced to insignificant levels, their generalization performance is solely determined by the statistical properties of their estimation procedure.

# 4  Conclusion

Taking in account budget constraints on both the number of examples and the computation time, we find *qualitative differences* between the generalization performance of small-scale learning systems and large-scale learning systems. The generalization properties of large-scale learning systems depend on both the statistical properties of the estimation procedure and the computational properties of the optimization algorithm. We illustrate this fact with some asymptotic results on gradient algorithms.

Considerable refinements of this framework can be expected. Extending the analysis to regularized risk formulations would make results on the complexity of primal and dual optimization algorithms [19, 20] directly exploitable. The choice of surrogate loss function [7, 12] could also have a non-trivial impact in the large-scale case.

**Acknowledgments**   Part of this work was funded by NSF grant CCR-0325463.

## Footnotes

[1]We often consider nested families of functions of the form $F_c = \{f \in \mathcal{H}, \; \Omega(f) \leq c\}$. Then, for each value of $c$, function $f_n$ is obtained by minimizing the regularized empirical risk $E_n(f) + \lambda\Omega(f)$ for a suitable choice of the Lagrange coefficient $\lambda$. We can then control the estimation-approximation tradeoff by choosing $\lambda$ instead of $c$.

[2]Although the original Vapnik-Chervonenkis bounds have the form $c\sqrt{\frac{d}{n}\log\frac{n}{d}}$, the logarithmic term can be eliminated using the "chaining" technique (e.g., [10].)

## References

[1] Leslie G. Valiant. A theory of learnable. *Proc. of the 1984 STOC*, pages 436–445, 1984.

[2] Vladimir N. Vapnik. *Estimation of Dependences Based on Empirical Data*. Springer Series in Statistics. Springer-Verlag, Berlin, 1982.

[3] Stéphane Boucheron, Olivier Bousquet, and Gábor Lugosi. Theory of classification: a survey of recent advances. *ESAIM: Probability and Statistics*, 9:323–375, 2005.

[4] Peter L. Bartlett and Shahar Mendelson. Empirical minimization. *Probability Theory and Related Fields*, 135(3):311–334, 2006.

[5] J. Stephen Judd. On the complexity of loading shallow neural networks. *Journal of Complexity*, 4(3):177–192, 1988.

[6] Richard O. Duda and Peter E. Hart. *Pattern Classification And Scene Analysis*. Wiley and Son, 1973.

[7] Tong Zhang. Statistical behavior and consistency of classification methods based on convex risk minimization. *The Annals of Statistics*, 32:56–85, 2004.

[8] Clint Scovel and Ingo Steinwart. Fast rates for support vector machines. In Peter Auer and Ron Meir, editors, *Proceedings of the 18th Conference on Learning Theory (COLT 2005)*, volume 3559 of *Lecture Notes in Computer Science*, pages 279–294, Bertinoro, Italy, June 2005. Springer-Verlag.

[9] Vladimir N. Vapnik, Esther Levin, and Yann LeCun. Measuring the VC-dimension of a learning machine. *Neural Computation*, 6(5):851–876, 1994.

[10] Olivier Bousquet. *Concentration Inequalities and Empirical Processes Theory Applied to the Analysis of Learning Algorithms*. PhD thesis, Ecole Polytechnique, 2002.

[11] Alexandre B. Tsybakov. Optimal aggregation of classifiers in statistical learning. *Annals of Statististics*, 32(1), 2004.

[12] Peter L. Bartlett, Michael I. Jordan, and Jon D. McAuliffe. Convexity, classification and risk bounds. *Journal of the American Statistical Association*, 101(473):138–156, March 2006.

[13] Pascal Massart. Some applications of concentration inequalities to statistics. *Annales de la Faculté des Sciences de Toulouse*, series 6, 9(2):245–303, 2000.

[14] Wee S. Lee, Peter L. Bartlett, and Robert C. Williamson. The importance of convexity in learning with squared loss. *IEEE Transactions on Information Theory*, 44(5):1974–1980, 1998.

[15] Shahar Mendelson. A few notes on statistical learning theory. In Shahar Mendelson and Alexander J. Smola, editors, *Advanced Lectures in Machine Learning*, volume 2600 of *Lecture Notes in Computer Science*, pages 1–40. Springer-Verlag, Berlin, 2003.

[16] John E. Dennis, Jr. and Robert B. Schnabel. *Numerical Methods For Unconstrained Optimization and Nonlinear Equations*. Prentice-Hall, Inc., Englewood Cliffs, New Jersey, 1983.

[17] Noboru Murata. A statistical study of on-line learning. In David Saad, editor, *Online Learning and Neural Networks*. Cambridge University Press, Cambridge, UK, 1998.

[18] Léon Bottou and Yann Le Cun. Large scale online learning. In Sebastian Thrun, Lawrence K. Saul, and Bernhard Schölkopf, editors, *Advances in Neural Information Processing Systems 16*. MIT Press, Cambridge, MA, 2004.

[19] Thorsten Joachims. Training linear SVMs in linear time. In *Proceedings of KDD'06*, Philadelphia, PA, USA, August 20-23 2006. ACM.

[20] Don Hush, Patrick Kelly, Clint Scovel, and Ingo Steinwart. QP algorithms with guaranteed accuracy and run time for support vector machines. *Journal of Machine Learning Research*, 7:733–769, 2006.
